# Receptive Fields without Spike-Triggering

**Jakob H Macke**

jakob@tuebingen.mpg.de

Max Planck Institute for Biological Cybernetics

Spemannstrasse 41

72076 Tübingen, Germany

**Günther Zeck**

zeck@neuro.mpg.de

Max Planck Institute of Neurobiology

Am Klopferspitze 18

82152 Martinsried, Germany

**Matthias Bethge**

mbethge@tuebingen.mpg.de

Max Planck Institute for Biological Cybernetics

Spemannstrasse 41

72076 Tübingen, Germany

## Abstract

Stimulus selectivity of sensory neurons is often characterized by estimating their receptive field properties such as orientation selectivity. Receptive fields are usually derived from the mean (or covariance) of the spike-triggered stimulus ensemble. This approach treats each spike as an independent message but does not take into account that information might be conveyed through patterns of neural activity that are distributed across space or time. Can we find a concise description for the processing of a whole population of neurons analogous to the receptive field for single neurons? Here, we present a generalization of the linear receptive field which is not bound to be triggered on individual spikes but can be meaningfully linked to distributed response patterns. More precisely, we seek to identify those stimulus features and the corresponding patterns of neural activity that are most reliably coupled. We use an extension of reverse-correlation methods based on canonical correlation analysis. The resulting *population receptive fields* span the subspace of stimuli that is most informative about the population response. We evaluate our approach using both neuronal models and multi-electrode recordings from rabbit retinal ganglion cells. We show how the model can be extended to capture nonlinear stimulus-response relationships using kernel canonical correlation analysis, which makes it possible to test different coding mechanisms. Our technique can also be used to calculate receptive fields from multi-dimensional neural measurements such as those obtained from dynamic imaging methods.

## 1  Introduction

Visual input to the retina consists of complex light intensity patterns. The interpretation of these patterns constitutes a challenging problem: for computational tasks like object recognition, it is not clear what information about the image should be extracted and in which format it should be represented. Similarly, it is difficult to assess what information is conveyed by the multitude of neurons in the visual pathway. Right from the first synapse, the information of an individual photoreceptor is signaled to many different cells with different temporal filtering properties, each of which is only a small unit within a complex neural network [20]. Even if we leave the difficulties imposed by nonlinearities and feedback aside, it is hard to judge what the contribution of any particular neuron is to the information transmitted.

The prevalent tool for characterizing the behavior of sensory neurons, the spike triggered average, is based on a quasi-linear model of neural responses [15]. For the sake of clarity, we consider an idealized model of the signaling channel

$$\mathbf{y} = W\mathbf{x} + \xi\,,\tag{1}$$

where $\mathbf{y} = (y_1, \ldots, y_N)^T$ denotes the vector of neural responses, $\mathbf{x}$ the stimulus parameters, $W = (\mathbf{w}_1, \ldots, \mathbf{w}_N)^T$ the filter matrix with row '$k$' containing the receptive field $\mathbf{w}_k$ of neuron $k$, and $\xi$ is the noise. The spike-triggered average only allows description of the stimulus-response function (i.e. the $\mathbf{w}_k$) of one single neuron at a time. In order to understand the collective behavior of a neuronal population, we rather have to understand the behavior of the matrix $W$, and the structure of the noise correlations $\Sigma_\xi$: Both of them influence the feature selectivity of the population.

Can we find a compact description of the features that a neural ensemble is most sensitive to? In the case of a single cell, the receptive field provides such a description: It can be interpreted as the "favorite stimulus" of the neuron, in the sense that the more similar an input is to the receptive field, the higher is the spiking probability, and thus the firing rate of the neuron. In addition, the receptive field can easily be estimated using a spike-triggered average, which, under certain assumptions, yields the optimal estimate of the receptive field in a linear-nonlinear cascade model [11].

If we are considering an ensemble of neurons rather than a single neuron, it is not obvious what to trigger on: This requires assumptions about what patterns of spikes or modulations in firing rates across the population carry information about the stimulus. Rather than addressing the question "*what* features of the stimulus are correlated with the occurence of spikes", the question now is: "*What* stimulus features are correlated with *what* patterns of spiking activity?" [14]. Phrased in the language of information theory, we are searching for the subspace that contains most of the mutual information between sensory inputs and neuronal responses. By this dimensionality reduction technique, we can find a compact description of the processing of the population.

As an efficient implementation of this strategy, we present an extension of reverse-correlation methods based on canonical correlation analysis. The resulting population receptive fields (PRFs) are not bound to be triggered on individual spikes but are linked to response patterns that are simultaneously determined by the algorithm.

We calculate the PRF for a population consisting of uniformly spaced cells with center-surround receptive fields and noise correlations, and estimate the PRF of a population of rabbit retinal ganglion cells from multi-electrode recordings. In addition, we show how our method can be extended to explore different hypotheses about the neural code, such as spike latencies or interval coding, which require nonlinear read out mechanisms.

## 2 From reverse correlation to canonical correlation

We regard the stimulus at time $t$ as a random variable $X_t \in \mathbb{R}^n$, and the neural response as $Y_t \in \mathbb{R}^m$. For simplicity, we assume that the stimulus consists of Gaussian white noise, i.e. $E(X) = 0$ and $Cov(X) = \mathbb{I}$.

The spike-triggered average $\mathbf{a}$ of a neuron can be motivated by the fact that it is the direction in stimulus-space maximizing the correlation-coefficient

$$\rho = \frac{Cov(\mathbf{a}^T X, Y_1)}{\sqrt{Var(\mathbf{a}^T X)Var(Y_1)}}.\tag{2}$$

between the filtered stimulus $\mathbf{a}^T X$ and a univariate neural response $Y_1$. In the case of a neural population, we are not only looking for the stimulus feature $\mathbf{a}$, but also need to determine what pattern of spiking activity $\mathbf{b}$ it is coupled with. The natural extension is to search for those vectors $\mathbf{a}_1$ and $\mathbf{b}_1$ that maximize

$$\rho_1 = \frac{Cov(\mathbf{a}_1^T X, \mathbf{b}_1^T Y)}{\sqrt{Var(\mathbf{a}_1^T X)Var(\mathbf{b}_1^T Y)}}.\tag{3}$$

We interpret $\mathbf{a}_1$ as the stimulus filter whose output is maximally correlated with the output of the "response filter" $\mathbf{b}_1$. Thus, we are simultaneously searching for features of the stimulus that the neural system is selective for, and the patterns of activity that it uses to signal the presence or absence

of this feature. We refer to the vector $\mathbf{a}_1$ as the (first) *population receptive field* of the population, and $\mathbf{b}_1$ is the *response feature* corresponding to $\mathbf{a}_1$. If a hypothetical neuron receives input from the population, and wants to decode the presence of the stimulus $\mathbf{a}_1$, the weights of the optimal linear readout [16] could be derived from $\mathbf{b}_1$.

Canonical Correlation Analysis (CCA) [9] is an algorithm that finds the vectors $\mathbf{a}_1$ and $\mathbf{b}_1$ that maximize (3): We denote the covariances of $X$ and $Y$ by $\Sigma_x$, $\Sigma_y$, the cross-covariance by $\Sigma_{xy}$, and the whitened cross-covariance by

$$C = \Sigma_x^{(-1/2)} \Sigma_{xy} \Sigma_y^{(-1/2)} \ . \tag{4}$$

Let $C = UDV^T$ denote the singular value decomposition of $C$, where the entries of the diagonal matrix $D$ are non-negative and decreasing along the diagonal. Then, the $k$-th pair of canonical variables is given by $\mathbf{a}_k = \Sigma_x^{(-1/2)} \mathbf{u}_k$ and $\mathbf{b}_k = \Sigma_y^{(-1/2)} \mathbf{v}_k$, where $\mathbf{u}_k$ and $\mathbf{v}_k$ are the $k$-th column vectors of $U$ and $V$, respectively. Furthermore, the $k$-th singular value of $C$, i.e. the $k$-th diagonal entry of $D$ is the correlation-coefficient $\rho_k$ of $\mathbf{a}_k^T X$ and $\mathbf{b}_k^T Y$. The random variables $\mathbf{a}_i^T X$ and $\mathbf{a}_j^T X$ are uncorrelated for $i \neq j$.

Importantly, the solution for the optimization problem in CCA is unique and can be computed efficiently via a single eigenvalue problem. The population receptive fields and the characteristic patterns are found by a joint optimization in stimulus and response space. Therefore, one does not need to know—or assume—a priori what features the population is sensitive to, or what spike patterns convey the information.

The first $K$ PRFs form a basis for the subspace of stimuli that the neural population is most sensitive to, and the individual basis vectors $\mathbf{a}_k$ are sorted according to their "informativeness" [13, 17].

The mutual information between two one-dimensional Gaussian Variables with correlation $\rho$ is given by $\mathrm{MI}_{Gauss} = -\frac{1}{2} \log(1 - \rho^2)$, so maximizing correlation coefficients is equivalent to maximizing mutual information [3]. Assuming the neural response $Y$ to be Gaussian, the subspace spanned by the first $K$ vectors $B_K = (\mathbf{b}_1, \ldots, \mathbf{b}_K)$ is also the $K$-subspace of stimuli that contains the maximal amount of mutual information between stimuli and neural response. That is

$$B_K = \operatorname*{argmax}_{B \in R^{n \times k}} \frac{\det\left(B^T \Sigma_y B\right)}{\det\left(B^T \left(\Sigma_y - \Sigma_{xy}^T \Sigma_x^{(-1)} \Sigma_{xy}\right) B\right)} \quad . \tag{5}$$

Thus, in terms of dimensionality reduction, CCA optimizes the same objective as *oriented PCA* [5]. In contrast to oriented PCA, however, CCA does not require one to know explicitly how the response covariance $\Sigma_y = \Sigma_s + \Sigma_\xi$ splits into signal $\Sigma_s$ and noise $\Sigma_\xi$ covariance. Instead, it uses the cross-covariance $\Sigma_{xy}$ which is directly available from reverse correlation experiments. In addition, CCA not only returns the most predictable response features $\mathbf{b}_1, \ldots \mathbf{b}_K$ but also the most predictive stimulus components $A_K = (\mathbf{a}_1, \ldots \mathbf{a}_K)$.

For general $Y$ and for stimuli $X$ with elliptically contoured distribution, $\mathrm{MI}_{Gauss} - J(A^T X)$ provides a lower bound to the mutual information between $A^T X$ and $B^T Y$, where

$$J(A^T X) = \frac{1}{2} \log(\det(2\pi e A^T \Sigma_x A)) - h(A^T X) \tag{6}$$

is the Negentropy of $A^T X$, and $h(A^T X)$ its differential entropy. Since for elliptically contoured distributions $J(A^T X)$ does not depend on A, the PRFs can be seen as the solution of a variational approach, maximizing a lower bound to the mutual information. Maximizing mutual information directly is hard, requires extensive amounts of data, and usually multiple repetitions of the same stimulus sequence.

## 3 The receptive field of a population of neurons

### 3.1 The effect of tuning functions and noise correlations

To illustrate the relationship between the tuning-functions of individual neurons and the PRFs [22], we calculate the first PRF of a simple one-dimensional population model consisting of center-

surround neurons. Each tuning function is modeled by a "Difference of Gaussians" (DOG)

$$f(x) = \exp\left(-\frac{1}{2}\left(\frac{x-c}{\sigma}\right)^2\right) - A\exp\left(-\frac{1}{2}\left(\frac{x-c}{\eta}\right)^2\right) \tag{7}$$

whose centers $c$ are uniformly distributed over the real axis. The width $\eta$ of the negative Gaussian is set to be twice as large as the width $\sigma$ of the positive Gaussian. If the area of both Gaussians is the same ($A = 1$), the DC component of the DOG-fillter is zero, i.e. the neuron is not sensitive to the mean luminance of the stimulus. If the ratio between both areas becomes substantially unbalanced, the DC component will become the largest signal ($A \approx 0$).

In addition to the parameter $A$, we will study the length scale of noise correlations $\lambda$ [18]. Specifically, we assume exponentially decaying noise correlation with $\Sigma_\xi(s) = \exp(-|s|/\lambda)$.

As this model is invariant under spatial shifts, the first PRF can be calculated by finding the spatial frequency at which the SNR is maximal. That is, the first PRF can be used to estimate the passband of the population transfer function. The SNR is given by

$$\text{SNR}(\omega) = \left(\frac{1+\lambda^2\omega^2}{2\lambda}\left(e^{-\omega^2\sigma^2} + A^2 e^{-\eta^2\omega^2} - 2A e^{-\frac{\sigma^2+\eta^2}{2}\omega^2}\right)\right)^2. \tag{8}$$

The passband of the first population filter moves as a function of both parameters $A$ and $\lambda$. It equals the DC component for small A (i.e. large imbalance) and small $\lambda$ (i.e. short correlation length). In this case, the mean intensity is the stimulus property that is most faithfully signaled by the ensemble.

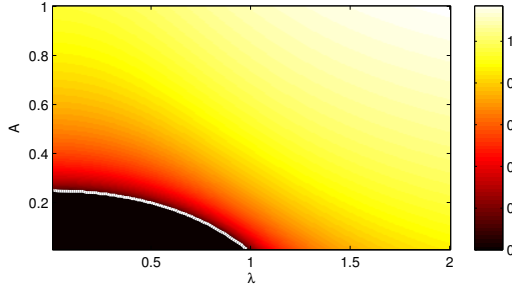

Figure 1: Spatial frequency of the first PRF for the model described above. $\lambda$ is the length-scale of the noise correlations, $A$ is the weight of the negative Gaussian in the DOG-model. The region in the bottom left corner (bounded by the white line) is the part of the parameter-space in which the PRF equals the DC component.

## 3.2 The receptive field of an ensemble of retinal ganglion cells

We mapped the population receptive fields of rabbit retinal ganglion cells recorded with a whole-mount preparation. We are not primarily interested in prediction performance [12], but rather in dimensionality reduction: We want to characterize the filtering properties of the population.

The neurons were stimulated with a $16 \times 16$ checkerboard consisting of binary white noise which was updated every 20ms. The experimental procedures are described in detail in [21]. After spike-sorting, spike trains from 32 neurons were binned at 20ms resolution, and the response of a neuron to a stimulus at time $t$ was defined to consist of the the spike-counts in the 10 bins between 40ms and 240ms after $t$. Thus, each population response $Y_t$ is a 320 dimensional vector.

Figure 3.2 A) displays the first 6 PRFs, the corresponding patterns of neural activity (B) and their correlation coefficients $\rho_k$ (which were calculated using a cross-validation procedure). It can be seen that the PRFs look very different to the usual center-surround structure of retinal ganglion. However, one should keep in mind that it is really the space spanned by the PRFs that is relevant, and thus be careful when interpreting the actual filter shapes [15].

For comparison, we also plotted the single-cell receptive fields in Figure 3.2 C), and their projections into the spaced spanned by the first 6 PRFs. These plots suggest that a small number of PRFs might

be sufficient to approximate each of the receptive fields. To determine the dimensionality of the relevant subspace, we analyzed the correlation-coefficients $\rho_k$. The Gaussian Mutual Information $MI_{Gauss} = -\frac{1}{2} \sum_{k=1}^{K} \log(1 - \rho_k^2)$ is an estimate of the information contained in the subspace spanned by the first $K$ PRFs. Based on this measure, a 12 dimensional subspace accounts for 90% of the total information.

In order to link the empirically estimated PRFs with the theoretical analysis in section 3.1, we calculated the spectral properties of the first PRF. Our analysis revealed that most of the power is in the low frequencies, suggesting that the population is in the parameter-regime where the single-cell receptive fields have power in the DC-component and the noise-correlations have short range, which is certainly reasonable for retinal ganglion cells [4].

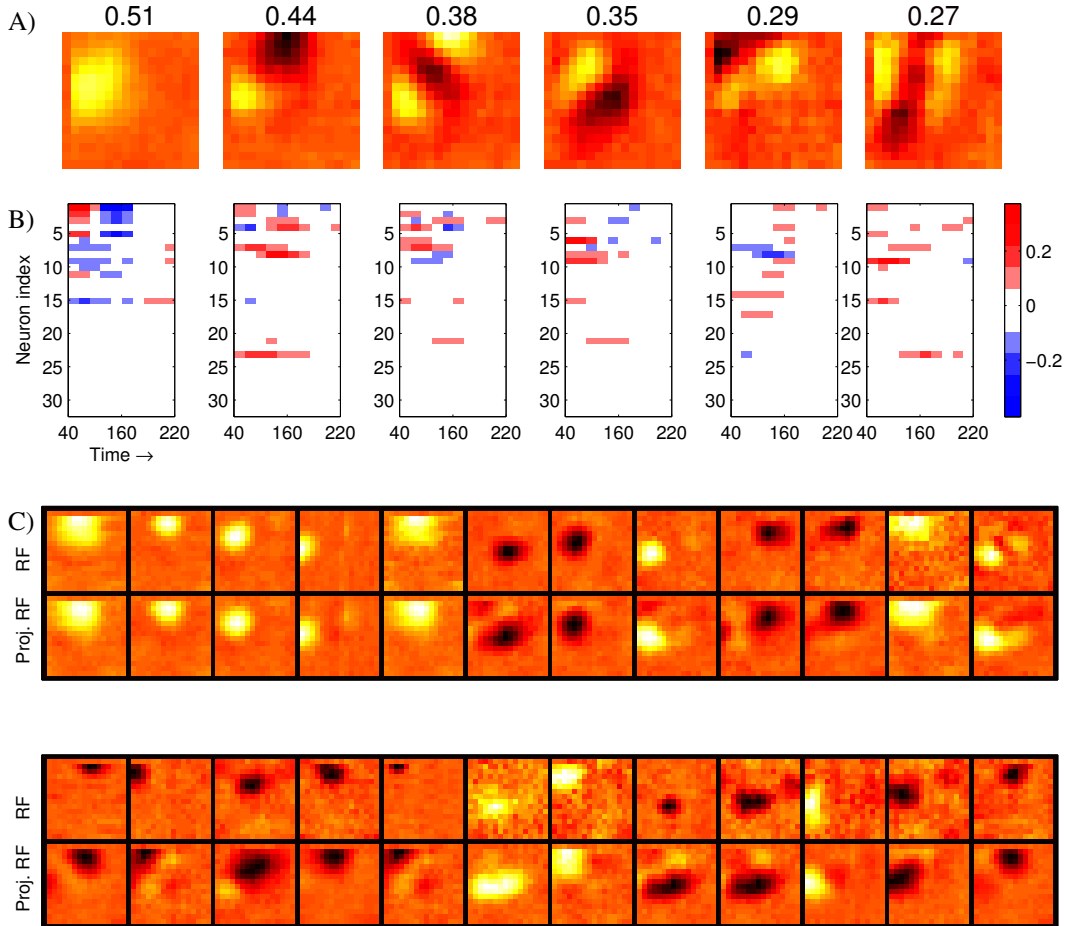

Figure 2: The population receptive fields of a group of 32 retinal ganglion cells: A) the first 6 PRFs, as sorted by the correlation coefficient $\rho_k$ B) the response features $b_k$ coupled with the PRFs. Each row of each image corresponds to one neuron, and each column to one time-bin. Blue color denotes enhanced activity, red suppressed. It can be seen that only a subset of neurons contributed to the first 6 PRFs. C) The single-cell receptive fields of 24 neurons from our population, and their projections into the space spanned by the 6 PRFs.

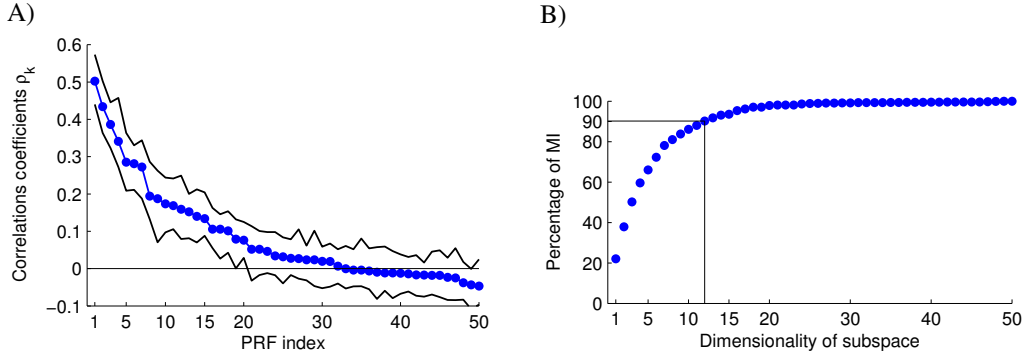

Figure 3: A) Correlation coefficients $\rho_k$ for the PRFs. Estimates and error-bars are calculated using a cross-validation procedure. B) Gaussian-MI of the subspace spanned by the first $K$ PRFs.

## 4 Nonlinear extensions using Kernel Canonical Correlation Analysis

Thus far, our model is completely linear: We assume that the stimulus is linearly related to the neural responses, and we also assume a linear readout of the response. In this section, we will explore generalizations of the CCA model using *Kernel CCA*: By embedding the stimulus-space nonlinearly in a feature space, nonlinear codes can be described.

Kernel methods provide a framework for extending linear algorithms to the nonlinear case [8]. After projecting the data into a feature space via a feature maps $\phi$ and $\psi$, a solution is found using linear methods in the feature space. In the case of Kernel CCA [1, 10, 2, 7] one seeks to find a linear relationship between the random variables $\hat{X} = \phi(X)$ and $\hat{Y} = \psi(Y)$, rather than between $X$ and $Y$. If an algorithm is purely defined in terms of dot-products, and if the dot-product in feature space $k(s,t) = \langle \psi(s), \psi(t) \rangle$ can be computed efficiently, then the algorithm does not require explicit calculation of the feature maps $\phi$ and $\psi$. This "kernel-trick" makes it possible to work in high- (or infinite)-dimensional feature spaces. It is worth mentioning that the space of patterns $Y$ itself does not have to be a vector space. Given a data-set $x_1 \ldots x_n$, it suffices to know the dot-products between any pair of training points, $K_{ij} := \langle \psi(y_i), \psi(y_j) \rangle$.

The kernel function $k(s,t)$ can be seen as a similiarity measure. It incorporates our assumptions about which spike-patterns should be regarded as similar "messages". Therefore, the choice of the kernel-function is closely related to specifing what the search-space of *potential* neural codes is. A number of distance- and kernel-functions [6, 19] have been proposed to compute distances between spike-trains. They can be designed to take into account precisely timed pattern of spikes, or to be invariant to certain transformations such as temporal jitter.

We illustrate the concept on simulated data: We will use a similarity measure based on the metric $D^{\text{interval}}$ [19] to estimate the receptive field of a neuron which does not use its firing rate, but rather the occurrence of specific interspike intervals to convey information about the stimulus. The metric $D^{\text{interval}}$ between two spike-trains is essentially the cost of matching their intervals by shifting, adding or deleting spikes. (We set $k(s,t) = \exp(-D(s,t)$. In theory, this function is not guaranteed to be positive definite, which could lead to numerical problems, but we did not encounter any in our simulation.) If we consider coding-schemes that are based on patterns of spikes, the methods described here become useful even for the analysis of single neurons. We will here concentrate on a single neuron, but the analysis can be extended to patterns distributed across several neurons.

Our hypothetical neuron encodes information in a pattern consisting of three spikes: The relative timing of the second spike is informative about the stimulus: The bigger the correlation between receptive field and stimulus $\langle \mathbf{r}, \mathbf{s}_t \rangle$, the shorter is the interval. If the receptive field is very dissimilar to the stimulus, the interval is long. While the timing of the spikes relative to each other is precise, there is jitter in the timing of the pattern relative to the stimulus. Figure 4 A) is a raster plot of simulated spike-trains from this model, ordered by $\langle \mathbf{r}, \mathbf{s}_t \rangle$. We also included noise spikes at random times.

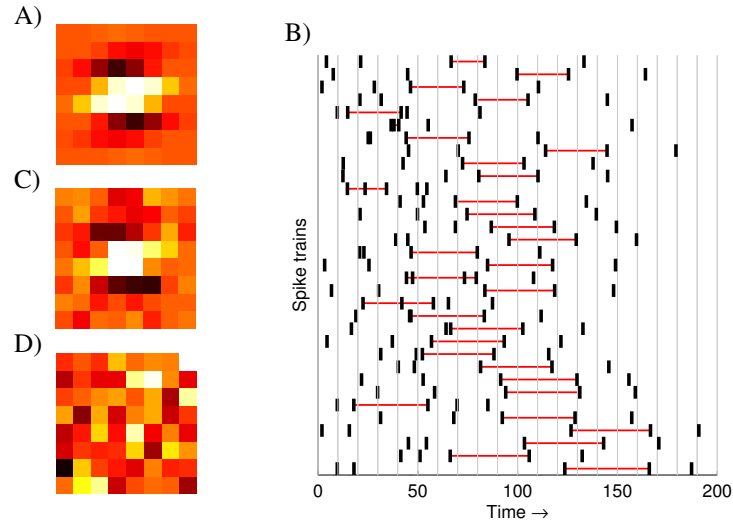

Figure 4: Coding by spike patterns: A) Receptive field of neuron described in Section 4. B) A subset of the simulated spike-trains, sorted with respect to the similarity between the shown stimulus and the receptive field of the model. The interval between the first two informative spikes in each trial is highlighted in red. C) Receptive field recovered by Kernel CCA, the correlation coefficient between real and estimated receptive field is 0.93. D) Receptive field derived using linear decoding, correlation coefficient is 0.02.

Using these spike-trains, we tried to recover the receptive field **r** without telling the algorithm what the indicating pattern was. Each stimulus was shown only once, and therefore, that every spike-pattern occurred only once. We simulated 5000 stimulus presentations for this model, and applied Kernel CCA with a linear kernel on the stimuli, and the alignment-score on the spike-trains. By using incomplete Cholesky decompositions [2], one can compute Kernel CCA without having to calculate the full kernel matrix. As many kernels on spike trains are computationally expensive, this trick can result in substantial speed-ups of the computation. The receptive field was recovered (see Figure 4), despite the highly nonlinear encoding mechanism of the neuron. For comparison, we also show what receptive field would be obtained using linear decoding on the indicated bins.

Although this neuron model may seem slightly contrived, it is a good proof of concept that, in principle, receptive fields can be estimated even if the firing rate gives no information at all about the stimulus, and the encoding is highly nonlinear. Our algorithm does not only look at patterns that occur more often than expected by chance, but also takes into account to what extent their occurrence is correlated to the sensory input.

## 5 Conclusions

We set out to find a useful description of the stimulus-response relationship of an ensemble of neurons akin to the concept of receptive field for single neurons. The population receptive fields are found by a joint optimization over stimuli and spike-patterns, and are thus not bound to be triggered by single spikes.

We estimated the PRFs of a group of retinal ganglion cells, and found that the first PRF had most spectral power in the low-frequency bands, consistent with our theoretical analysis. The stimulus we used was a white-noise sequence—it will be interesting to see how the informative subspace and its spectral properties change for different stimuli such as colored noise. The ganglion cell layer of the retina is a system that is relatively well understood at the level of single neurons. Therefore, our results can readily be compared and connected to those obtained using conventional analysis techniques. However, our approach has the potential to be especially useful in systems in which the functional significance of single cell receptive fields is difficult to interpret.

We usually assumed that each dimension of the response vector $Y$ represents an electrode-recording from a single neuron. However, the vector $Y$ could also represent any other multi-dimensional measurement of brain activity: For example, imaging modalities such as voltage-sensitive dye imaging yield measurements at multiple pixels simultaneously. Data from electro-physiological data, e.g. local field potentials, are often analyzed in frequency space, i.e. by looking at the energy of the signal in different frequency bands. This also results in a multi-dimensional representation of the signal. Using CCA, receptive fields can readily be estimated from these kinds of representations without limiting attention to single channels or extracting neural events.

### Acknowledgments

We would like to thank A Gretton and J Eichhorn for useful discussions, and F Jäkel, J Butler and S Liebe for comments on the manuscript.

## References

[1] S. Akaho. A kernel method for canonical correlation analysis. In *International Meeting of Psychometric Society, Osaka*, 2001.

[2] F. R. Bach and M. I. Jordan. Kernel independent component analysis. *Journal of Machine Learning Research*, 3:1:48, 2002.

[3] G. Chechik, A. Globerson, N. Tishby, and Y. Weiss. Information Bottleneck for Gaussian Variables. *The Journal of Machine Learning Research*, 6:165–188, 2005.

[4] S. Devries and D. Baylor. Mosaic Arrangement of Ganglion Cell Receptive Fields in Rabbit Retina. *Journal of Neurophysiology*, 78(4):2048–2060, 1997.

[5] K. Diamantaras and S. Kung. Cross-correlation neural network models. *Signal Processing, IEEE Transactions on*, 42(11):3218–3223, 1994.

[6] J. Eichhorn, A. Tolias, A. Zien, M. Kuss, C. E. Rasmussen, J. Weston, N. Logothetis, and B. Schölkopf. Prediction on spike data using kernel algorithms. In S. Thrun, L. Saul, and B. Schölkopf, editors, *Advances in Neural Information Processing Systems 16*. MIT Press, Cambridge, MA, 2004.

[7] K. Fukumizu, F. R. Bach, and A. Gretton. Statistical consistency of kernel canonical correlation analysis. *Journal of Machine Learning Research*, 2007.

[8] T. Hofmann, B. Schölkopf, and A. Smola. Kernel methods in machine learning. *Annals of Statistics (in press)*, 2007.

[9] H. Hotelling. Relations between two sets of variates. *Biometrika*, 28:321–377, 1936.

[10] T. Melzer, M. Reiter, and H. Bischof. Nonlinear feature extraction using generalized canonical correlation analysis. In *Proc. of International Conference on Artificial Neural Networks (ICANN)*, pages 353–360, 8 2001.

[11] L. Paninski. Convergence properties of three spike-triggered analysis techniques. *Network*, 14(3):437–64, Aug 2003.

[12] J. W. Pillow, L. Paninski, V. J. Uzzell, E. P. Simoncelli, and E. J. Chichilnisky. Prediction and decoding of retinal ganglion cell responses with a probabilistic spiking model. *J Neurosci*, 25(47):11003–13, 2005.

[13] J. W. Pillow and E. P. Simoncelli. Dimensionality reduction in neural models: an information-theoretic generalization of spike-triggered average and covariance analysis. *J Vis*, 6(4):414–28, 2006.

[14] M. J. Schnitzer and M. Meister. Multineuronal firing patterns in the signal from eye to brain. *Neuron*, 37(3):499–511, 2003.

[15] O. Schwartz, J. W. Pillow, N. C. Rust, and E. P. Simoncelli. Spike-triggered neural characterization. *J Vis*, 6(4):484–507, 2006.

[16] H. S. Seung and H. Sompolinsky. Simple models for reading neuronal population codes. *Proc Natl Acad Sci U S A*, 90(22):10749–53, 1993.

[17] T. Sharpee, N. Rust, and W. Bialek. Analyzing neural responses to natural signals: maximally informative dimensions. *Neural Comput*, 16(2):223–50, 2004.

[18] H. Sompolinsky, H. Yoon, K. Kang, and M. Shamir. Population coding in neuronal systems with correlated noise. *Phys Rev E Stat Nonlin Soft Matter Phys*, 64(5 Pt 1):051904, 2001.

[19] J. Victor. Spike train metrics. *Curr Opin Neurobiol*, 15(5):585–92, 2005.

[20] H. Wässle. Parallel processing in the mammalian retina. *Nat Rev Neurosci*, 5(10):747–57, 2004.

[21] G. M. Zeck, Q. Xiao, and R. H. Masland. The spatial filtering properties of local edge detectors and brisk-sustained retinal ganglion cells. *Eur J Neurosci*, 22(8):2016–26, 2005.

[22] K. Zhang and T. Sejnowski. Neuronal Tuning: To Sharpen or Broaden?, 1999.

